# Multiple-Instance Active Learning

**Burr Settles   Mark Craven**
University of Wisconsin
Madison, WI 5713 USA
{bsettles@cs,craven@biostat}.wisc.edu

**Soumya Ray**
Oregon State University
Corvallis, OR 97331 USA
sray@eecs.oregonstate.edu

## Abstract

We present a framework for active learning in the *multiple-instance* (MI) setting. In an MI learning problem, instances are naturally organized into bags and it is the bags, instead of individual instances, that are labeled for training. MI learners assume that every instance in a bag labeled *negative* is actually negative, whereas at least one instance in a bag labeled *positive* is actually positive. We consider the particular case in which an MI learner is allowed to selectively query unlabeled instances from positive bags. This approach is well motivated in domains in which it is inexpensive to acquire bag labels and possible, but expensive, to acquire instance labels. We describe a method for learning from labels at mixed levels of granularity, and introduce two active query selection strategies motivated by the MI setting. Our experiments show that learning from instance labels can significantly improve performance of a basic MI learning algorithm in two multiple-instance domains: content-based image retrieval and text classification.

## 1   Introduction

A limitation of supervised learning is that it requires a set of instance labels which are often difficult or expensive to obtain. The *multiple-instance* (MI) learning framework [3] can, in some cases, address this handicap by relaxing the granularity at which labels are given. In the MI setting, instances are grouped into *bags* (i.e., multi-sets) which may contain any number of instances. A bag is labeled *negative* if and only if it contains all negative instances. A bag is labeled *positive*, however, if at least one of its instances is positive. Note that positive bags may also contain negative instances. The MI setting was formalized by Dietterich et al. in the context of drug activity prediction [3], and has since been applied to a wide variety of tasks including content-based image retrieval [1, 6, 8], text classification [1, 9], stock prediction [6], and protein family modeling [10].

Figure 1 illustrates how the MI representation can be applied to (a) content-based image retrieval (CBIR) and (b) text classification tasks. For the CBIR task, images are represented as bags and instances correspond to segmented regions of the image. A bag representing a given image is labeled positive if the image contains some object of interest. The multiple-instance paradigm is well suited to this task because only a few regions of an image may represent the object of interest, such as the gold medal in Figure 1(a). An advantage of the MI representation here is that it is significantly easier to label an entire image than it is to label each segment. For text classification, documents are represented as bags and instances correspond to short passages (e.g., paragraphs) in the documents. This formulation is useful in classification tasks for which document labels are freely available or cheaply obtained, but the target concept is represented by only a few passages. For example, consider the task of classifying articles according whether or not they contain information about the sub-cellular location of proteins. The article in Figure 1(b) is labeled by the Mouse Genome Database [4] as a citation for the protein *catalase* that specifies its sub-cellular location. However, the text that states this is only a short passage on the second page of the article. The MI approach is therefore compelling because document labels can be cheaply obtained (say from the Mouse Genome Database), but the labeling is not readily available at the most appropriate level of granularity (passages).

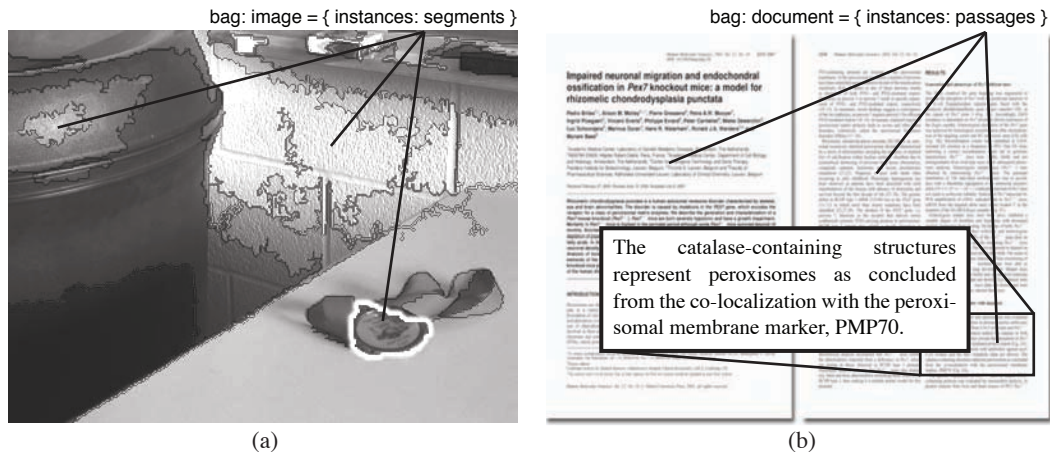

bag: image = { instances: segments }

bag: document = { instances: passages }

The catalase-containing structures represent peroxisomes as concluded from the co-localization with the peroxisomal membrane marker, PMP70.

(a)                                                    (b)

Figure 1: Motivating examples for multiple-instance active learning. (a) In content-based image retrieval, images are represented as bags and instances correspond to segmented image regions. An active MI learner may query which segments belong to the object of interest, such as the gold medal shown in this image. (b) In text classification, documents are bags and the instances represent passages of text. In MI active learning, the learner may query specific passages to determine if they are representative of the positive class at hand.

The main challenge of multiple-instance learning is that, to induce an accurate model of the target concept, the learner must determine which instances in positive bags are actually positive, even though the ratio of negatives to positives in these bags can be arbitrarily high. For many MI problems, such as the tasks illustrated in Figure 1, it is possible to obtain labels both at the bag level and directly at the instance level. Fully labeling all instances, however, is expensive. As mentioned above, the rationale for formulating the learning task as an MI problem is that it allows us to take advantage of coarse labelings that may be available at low cost, or even for free. The approach that we consider here is one that involves selectively obtaining the labels of certain instances in the context of MI learning. In particular, we consider obtaining labels for selected instances in positive bags, since the labels for instances in negative bags are known.

In *active learning* [2], the learner is allowed to ask *queries* about unlabeled instances. In this way, the *oracle* (or human annotator) is required to label only instances that are assumed to be most valuable for training. In the standard supervised setting, *pool-based* active learning typically begins with an initial learner trained with a small set of labeled instances. Then the learner can query instances from a large pool of unlabeled instances, re-train, and repeat. The goal is to reduce the total amount of labeling effort required for the learner to achieve a certain level of accuracy.

We argue that whereas multiple-instance learning reduces the burden of labeling data by getting labels at a coarse level of granularity, we may also benefit from selectively labeling some part of the training data at a finer level of granularity. Hence, we explore the approach of *multiple-instance active learning* as a way to efficiently overcome the ambiguity of the MI framework while keeping labeling costs low.

There are several MI active learning scenarios we might consider. The first, which is analogous to standard supervised active learning, is simply to allow the learner to query for the labels of unlabeled bags. A second scenario is one in which all bags in the training set are labeled and the learner is allowed to query for the labels of selected instances from positive bags. For example, the learner might query on particular image segments or passages of text in the CBIR and text classification domains, respectively. If an instance-query result is positive, the learner now has direct evidence for the positive class. If the query result is negative, the learner knows to focus its attention to other instances from that bag, also reducing ambiguity. A third scenario involves querying selected positive bags rather than instances, and obtaining labels for any (or all) instances in such bags. For example, the learner might query a positive image in the CBIR domain, and ask the oracle to label as many segments as desired. A final scenario would assume that some bags are labeled and some are not, and the learner would be able to query on (i) unlabeled bags, (ii) unlabeled instances in positive

bags, or (iii) some combination thereof. In the present work, we focus on the second formulation above, where the learner queries selected unlabeled instances from labeled, positive bags.

The rest of this paper is organized as follows. First, we describe the algorithms we use to train MI classifiers and select instance queries for active learning. Then, we describe our experiments to evaluate these approaches on two data sets in the CBIR and text classification domains. Finally, we discuss the results of our experiments and offer some concluding remarks.

## 2   Algorithms

**MI Logistic Regression.** We train probabilistic models for multiple-instance tasks using a generalization of the Diverse Density framework [6]. For MI classification, we seek the conditional probability that the label $y_i$ is positive for bag $B_i$ given $n$ constituent instances: $P(y_i = 1 | B_i = \{B_{i1}, B_{i2}, \ldots, B_{in}\})$. If a classifier can provide an equivalent probability $P(y_{ij} = 1 | B_{ij})$ for instance $B_{ij}$, we can use a *combining function* (such as softmax or noisy-or) to combine posterior probabilities of all the instances in a bag and estimate its posterior probability $P(y_i = 1 | B_i)$. The combining function here explicitly encodes the MI assumption. If the model finds an instance likely to be positive, the output of the combining function should find its corresponding bag likely to be positive as well.

In our work, we train classifiers using *multiple-instance logistic regression* (MILR) which has been shown to be a state-of-the-art MI learning algorithm, and appears to be a competitive method for text classification and CBIR tasks [9]. MILR uses logistic regression with parameters $\theta = (\mathbf{w}, b)$ to estimate conditional probabilities for each instance:

$$o_{ij} = P(y_{ij} = 1 | B_{ij}) = \frac{1}{1 + e^{-(\mathbf{w} \cdot B_{ij} + b)}}.$$

Here $B_{ij}$ represents a vector of feature values representing the $j$th instance in the $i$th bag, and $\mathbf{w}$ is a vector of weights associated with the features. In order to combine these class probabilities for instances into a class probability for a bag, MILR uses the softmax function:

$$o_i = P(y_i = 1 | B_i) = \text{softmax}_\alpha(o_{i1}, \ldots, o_{in}) = \frac{\sum_{j=1}^n o_{ij} e^{\alpha o_{ij}}}{\sum_{j=1}^n e^{\alpha o_{ij}}},$$

where $\alpha$ is a constant that determines the extent to which softmax approximates a hard max function.

In the general MI setting we do not know the labels of instances in positive bags. Because the equations above represent smooth functions of the model parameters $\theta$, however, we can learn parameter values using a gradient-based optimization method and an appropriate objective function. In the present work, we minimize squared error over the bags $E(\theta) = \frac{1}{2} \sum_i (y_i - o_i)^2$, where $y_i \in \{0, 1\}$ is the known label of bag $B_i$. While we describe our MI active learning methods below in terms of this formulation of MILR, it is important to note that they generalize to any classifier that outputs instance-level probabilities used with differentiable combining and objective functions. Diverse Density [6], for example, couples a Gaussian instance model with a noisy-or combining function.

**Learning from Labels at Mixed Granularities.** Suppose our active MI learner queries instance $B_{ij}$ and the corresponding instance label $y_{ij}$ is provided by the oracle. We would like to include a direct training signal for this instance in the optimization procedure above. However, $E(\theta)$ is defined in terms of bag-level error, not instance-level error. Consider, though, that in MI learning a labeled instance is effectively the same as a labeled bag that contains only that instance. So when the label for instance $B_{ij}$ is known, we transform the training set for each query by adding a new training tuple $\langle \{B_{ij}\}, y_{ij} \rangle$, where $\{B_{ij}\}$ is a new singleton bag containing only a copy of the queried instance, and $y_{ij}$ is the corresponding label. A copy of the query instance $B_{ij}$ also remains in the original bag $B_i$, enabling the learner to compute the remaining instance gradients as described below.

Since the objective function will guide the learner toward classifying the singleton query instance $B_{ij}$ in the positive tuple $\langle \{B_{ij}\}, 1 \rangle$ as positive, it will tend to classify the original bag $B_i$ positive as well. Conversely, if we add the negative tuple $\langle \{B_{ij}\}, 0 \rangle$, the learner will tend to classify the instance

negative in the original bag, which will affect the other instance gradients via the combining function and guides the learner to focus on other potentially positive instances in that bag.

It may seem that this effect on the original bag could be achieved by clamping the instance output $o_{ij}$ to $y_{ij}$ during training, but this has the undesirable property of eliminating the training signal for the bag and the instance. If $y_{ij} = 1$, the combining function output would be extremely high, making bag error nearly zero, thus minimizing the objective function without any actual parameter updates. If $y_{ij} = 0$, the instance would output nothing to the combining function, thus the learner would get no training signal for this instance (though in this case the learner can still focus on other instances in the bag). It is possible to combine clamped instance outputs with our singleton bag approach to overcome this problem, but our experiments indicate that this has no practical advantage over adding singleton bags alone.

Also note that simply adding singleton bags will alter the objective function by adding weight, albeit indirectly, to bags that have been queried more often. To control this effect, we uniformly weight each bag and all its queried singleton bags to sum to 1 when computing the value and gradient for the objective function during training. For example, an unqueried bag has weight 1, a bag with one instance query and its derived singleton bag each have weight 0.5, and so on.

**Uncertainty Sampling.** Now we turn our attention to strategies for selecting query instances for labeling. A common approach to active learning in the standard supervised setting is *uncertainty sampling* [5]. For probabilistic classifiers, this involves applying the classifier to each unlabeled instance and querying those with most uncertainty about the class label. Recall that the learned model estimates $o_{ij} = P(y_{ij} = 1 | B_{ij})$, the probability that instance $B_{ij}$ is positive. We represent the uncertainty $U(B_{ij})$ by the Gini measure:

$$U(B_{ij}) = 2o_{ij}(1 - o_{ij}).$$

Note that the particular measure we use here is not critical; the important properties are that its minima are at zero and one, its maximum is at 0.5, and it is symmetric about 0.5.

**MI Uncertainty (MIU).** We argue that when doing active learning in a multiple-instance setting, the selection criterion should take into account not just uncertainty about a given instance's class label, but also the extent to which the learner can adequately "explain" the bag to which the instance belongs. For example, the instance that the learner finds most uncertain may belong to the same bag as the instance it finds most positive. In this case, the learned model will have a high value of $P(y_i = 1 | B_i)$ for the bag because the value computed by the combining function will be dominated by the output of the positive-looking instance. We propose an uncertainty-based query strategy that weights the uncertainty of $B_{ij}$ in terms of how much it contributes to the classification of bag $B_i$. As such, we define the *MI Uncertainty* (MIU) of an instance to be the derivative of bag output with respect to instance output (i.e., the derivative of the softmax combining function) times instance uncertainty:

$$MIU(B_{ij}) = \frac{\partial o_i}{\partial o_{ij}} U(B_{ij}).$$

**Expected Gradient Length (EGL).** Another query strategy we consider is to identify the instance that would impart the greatest change to the current model if we knew its label. Since we train MILR with gradient descent, this involves querying the instance which, if $\langle \{B_{ij}\}, y_{ij} \rangle$ is added to the training set, would create the greatest change in the gradient of the objective function (i.e., the largest gradient vector used to re-estimate values for $\theta$). Let $\nabla E(\theta)$ be the gradient of $E$ with respect to $\theta$, which is a vector whose components are the partial derivatives of $E$ with respect to each model parameter: $\nabla E(\theta) = [\frac{\partial E}{\partial \theta_1}, \frac{\partial E}{\partial \theta_2}, \dots, \frac{\partial E}{\partial \theta_m}]$.

Now let $\nabla E_{ij}^+(\theta)$ be the new gradient obtained by adding the positive tuple $\langle \{B_{ij}\}, 1 \rangle$ to the training set, and likewise let $\nabla E_{ij}^-(\theta)$ be the new gradient if a query results in the negative tuple $\langle \{B_{ij}\}, 0 \rangle$ being added. Since we do not know which label the oracle will provide in advance, we instead calculate the *expected* length of the gradient based on the learner's current belief $o_{ij}$ in each outcome. More precisely, we define the *Expected Gradient Length* (EGL) to be:

$$EGL(B_{ij}) = o_{ij} \| \nabla E_{ij}^+(\theta) \| + (1 - o_{ij}) \| \nabla E_{ij}^-(\theta) \|.$$

Note that this selection strategy does not explicitly encode the MI bias. Instead, it employs class probabilities to determine the expected label for candidate queries, with the goal of maximizing parameter changes to what happens to be an MI learning algorithm. This strategy can be generalized to query for other properties in non-MI active learning as well. For example, Zhu et al. [11] use a related approach to determine the expected label of candidate query instances when combining active learning with graph-based semi-supervised learning. Rather than trying to maximize the expected change in the learning model, however, they select for the expected reduction in estimated error over unlabeled instances.

## 3 Data and Experiments

Since no MI data sets with instance-level labels previously existed, we augmented an existing MI data set by manually adding instance labels. SIVAL[1] is a collection for content-based image retrieval that includes 1500 images, each labeled with one of 25 class labels. The images contain complex objects photographed in a variety of positions, orientations, locations, and lighting conditions. The images (bags) have been transformed and segmented into approximately 30 segments (instances) each. Each segment is represented by a 30-dimensional feature vector describing color and texture attributes of the segment and its neighbors. For more details, see Rahmani & Goldman [8]. We modified the collection by manually annotating the instance segments that belong to the labeled object for each image using a graphical interface we developed.

We also created a semi-synthetic MI data set for text classification, using the 20 Newsgroups[2] corpus as a base. This corpus was chosen because it is an established benchmark for text classification, and because the source texts—newsnet posts from the early 1990s—are relatively short (in the MI setting, instances are usually paragraphs or short passages [1, 9]). For each of the 20 news categories, we generate artificial bags of approximately 50 posts (instances) each by randomly sampling from the target class (i.e., newsgroup category) at a rate of 3% for positive bags, with remaining instances (and all instances for negative bags) drawn uniformly from the other classes. The texts are processed with stemming, stop-word removal, and information-gain ranked feature selection. The TFIDF values of the top 200 features are used to represent the instance texts. We construct a data set of 100 bags (50 positives and 50 negatives) for each class.

We compare our *MI Uncertainty* (MIU) and *Expected Gradient Length* (EGL) selection strategies from Section 2 against two baselines: *Uncertainty* (using only the instance-model's uncertainty), and instances chosen uniformly at *Random* from positive bags (to evaluate the advantage of "passively" labeling instances). The MILR model uses $\alpha = 2.5$ for the softmax function and is trained by minimizing squared loss via L-BFGS [7]. The instance-labeled MI data sets and MI learning source code used in these experiments are available online[3].

We evaluate our methods by constructing learning curves that plot the area under the ROC curve (AUROC) as a function of instances queried for each data set and selection strategy. The initial point in all experiments is the AUROC for a model trained on labeled bags from the training set without any instance queries. Following previous work on the CBIR problem [8], we average results for SIVAL over 20 independent runs for each image class, where the learner begins with 20 randomly drawn positive bags (from which instances may be queried) and 20 random negative bags. The model is then evaluated on the remainder of the unlabeled bags, and labeled query instances are added to the training set in batches of size $q = 2$. For 20 Newsgroups, we average results using 10-fold cross-validation for each newsgroup category, using a query batch size of $q = 5$.

Due to lack of space, we cannot show learning curves for every task. Figure 2 shows three representative learning curves for each of the two data sets. In Table 1 we summarize all curves by reporting the average improvement made by each query selection strategy over the initial MILR model (before any instance queries) for various points along the learning curve. Table 2 presents a more detailed comparison of the initial model against each query selection method at a fixed point early on in active learning (10 query batches).

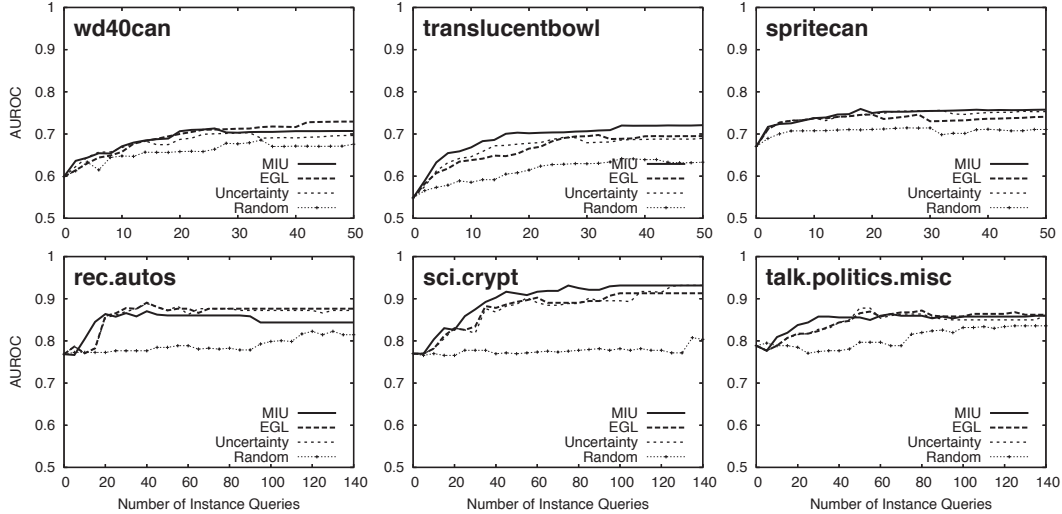

Figure 2: Sample learning curves from SIVAL (top row) and 20 Newsgroups (bottom row) tasks.

Table 1: Summary of learning curves. The average AUROC improvement over the initial MI model (before any instance queries) is reported for each selection strategy. Numbers are averaged across all tasks in each data set at various points during active learning. The winning algorithm at each point is indicated with a box.

| Instance Queries | SIVAL Tasks | | | | 20 Newsgroups Tasks | | | |
|---|---|---|---|---|---|---|---|---|
| | Random | Uncert. | EGL | MIU | Random | Uncert. | EGL | MIU |
| 10 | +0.023 | +0.043 | +0.039 | +0.050 | -0.001 | +0.002 | +0.002 | +0.009 |
| 20 | +0.033 | +0.065 | +0.063 | +0.070 | -0.002 | +0.015 | +0.015 | +0.029 |
| 50 | +0.057 | +0.084 | +0.085 | +0.087 | +0.002 | +0.046 | +0.045 | +0.051 |
| 80 | +0.065 | +0.088 | +0.093 | +0.090 | +0.003 | +0.052 | +0.056 | +0.056 |
| 100 | +0.068 | +0.092 | +0.095 | +0.090 | +0.008 | +0.055 | +0.055 | +0.058 |

## 4 Discussion of Results

We can draw several interesting conclusions from these results. First and most germane to MI active learning is that MI learners benefit from instance-level labels. With the exception of random selection on 20 Newsgroups data, instance-level labels almost always improve the accuracy of the learner, often with statistical significance after only a few queries.

Second, we see that active query strategies (e.g., Uncertainty, EGL, and MIU) perform better than passive (random) instance labeling. On SIVAL tasks, random querying steadily improves accuracy, but very slowly. As Table 1 shows, random selection at 100 queries fails to be competitive with the three active query strategies after half as many queries. On 20 Newsgroups tasks, random selection has a slight negative effect (if any) early on, possibly because it lacks a focused search for positive instances (of which there are only one or two per bag). All three active selection methods, on the other hand, show significant gains fairly quickly on both data sets.

Finally, MIU appears to be a well-suited query strategy for this formulation of MI active learning. On both data sets, it consistently improves the initial MI learner, usually with statistical significance, and often approaches the asymptotic level of accuracy with fewer labeled instances than the other two active methods. Uncertainty and EGL seem to perform quite comparably, with EGL performing slightly better between the two. MIU's gains over these other query strategies are not usually statistically significant, however, and in the long run it is generally matched or slightly surpassed by them. MIU shows the greatest advantage early in the active instance-querying process, perhaps because it is the only method we tested that explicitly encodes the MI assumption by taking advantage of the combining function in its estimation of value to the learner.

Table 2: Detailed comparison of the initial MI learner against various query strategies after 10 query batches (20 instances for SIVAL, 50 instances for 20 Newsgroups). Average AUROC values are shown for each algorithm on each task. Statistically significant gains over the initial learner (using a two-tailed $t$-test at 95%) are shown in bold. The winning algorithm for each task is indicated with a box, and a tally of wins for each algorithm is reported below each column.

| Task | Initial | Random | Uncert. | EGL | MIU |
|---|---|---|---|---|---|
| ajaxorange | 0.547 | 0.564 | **0.633** | [**0.638**] | **0.627** |
| apple | 0.431 | 0.418 | [**0.469**] | 0.455 | 0.459 |
| banana | 0.440 | 0.463 | [**0.514**] | **0.511** | **0.507** |
| bluescrunge | 0.410 | 0.426 | [**0.508**] | **0.470** | **0.491** |
| candlewithholder | 0.623 | **0.662** | 0.646 | **0.656** | [**0.677**] |
| cardboardbox | 0.430 | 0.437 | **0.451** | 0.442 | [**0.454**] |
| checkeredscarf | 0.662 | **0.749** | **0.765** | [**0.772**] | **0.765** |
| cokecan | 0.668 | **0.727** | 0.693 | **0.713** | [**0.736**] |
| dataminingbook | 0.445 | **0.480** | **0.505** | [**0.522**] | **0.519** |
| dirtyrunningshoe | 0.620 | **0.701** | **0.703** | **0.697** | [**0.708**] |
| dirtyworkgloves | 0.455 | [**0.497**] | **0.491** | **0.496** | [**0.497**] |
| fabricsoftenerbox | 0.417 | **0.534** | **0.617** | **0.594** | [**0.634**] |
| feltflowerrug | 0.743 | 0.754 | **0.794** | [**0.799**] | **0.792** |
| glazedwoodpot | 0.444 | **0.464** | [**0.528**] | **0.515** | **0.526** |
| goldmedal | 0.496 | **0.544** | [**0.622**] | **0.602** | **0.605** |
| greenteabox | 0.563 | **0.595** | **0.614** | **0.619** | [**0.639**] |
| juliespot | 0.479 | 0.490 | **0.571** | [**0.580**] | **0.564** |
| largespoon | [0.436] | 0.403 | 0.406 | 0.394 | 0.408 |
| rapbook | [0.478] | 0.455 | 0.463 | 0.454 | 0.457 |
| smileyfacedoll | 0.556 | **0.612** | [**0.675**] | **0.640** | **0.655** |
| spritecan | 0.670 | **0.711** | **0.749** | **0.746** | [**0.750**] |
| stripednotebook | 0.477 | 0.478 | 0.486 | [**0.519**] | 0.489 |
| translucentbowl | 0.548 | **0.614** | **0.678** | **0.665** | [**0.702**] |
| wd40can | 0.599 | **0.658** | **0.687** | **0.700** | [**0.707**] |
| woodrollingpin | 0.416 | [**0.435**] | 0.420 | 0.426 | 0.429 |
| | | | | | |
| alt.atheism | 0.812 | 0.836 | 0.863 | 0.839 | [0.877] |
| comp.graphics | 0.720 | 0.690 | 0.789 | 0.783 | [**0.819**] |
| comp.os.ms-windows.misc | [0.772] | 0.768 | 0.764 | 0.742 | 0.714 |
| comp.sys.ibm.pc.hardware | [0.716] | 0.690 | 0.687 | 0.694 | 0.707 |
| comp.sys.mac.hardware | 0.716 | 0.728 | **0.861** | **0.855** | [**0.878**] |
| comp.windows.x | 0.835 | 0.827 | 0.888 | [0.894] | 0.882 |
| misc.forsale | 0.769 | 0.748 | 0.758 | [0.777] | 0.771 |
| rec.autos | 0.768 | 0.785 | [**0.872**] | [**0.872**] | **0.860** |
| rec.motorcycles | 0.844 | 0.844 | 0.871 | 0.879 | [0.883] |
| rec.sport.baseball | 0.838 | 0.846 | 0.871 | 0.869 | [0.899] |
| rec.sport.hockey | 0.918 | 0.918 | [0.966] | 0.962 | 0.964 |
| sci.crypt | 0.770 | 0.770 | **0.887** | **0.893** | [**0.913**] |
| sci.electronics | 0.719 | [0.751] | 0.731 | 0.733 | 0.725 |
| sci.med | 0.827 | 0.819 | 0.837 | 0.845 | [0.862] |
| sci.space | 0.822 | 0.824 | **0.901** | [**0.905**] | **0.893** |
| soc.religion.christian | 0.768 | 0.780 | 0.769 | 0.771 | [0.789] |
| talk.politics.guns | 0.847 | 0.855 | 0.860 | [0.870] | 0.858 |
| talk.politics.mideast | 0.791 | 0.793 | 0.874 | [0.880] | 0.876 |
| talk.politics.misc | 0.789 | 0.797 | [**0.878**] | 0.866 | 0.856 |
| talk.religion.misc | 0.759 | 0.773 | 0.785 | 0.773 | [0.793] |
| | | | | | |
| TOTAL NUMBER OF WINS | 4 | 3 | 9 | 12 | 19 |

It is also interesting to note that in an earlier version of our learning algorithm, we did not normalize weights for bags and instance-query singleton bags when learning with labels at mixed granularities. Instead, all such bags were weighted equally and the objective function was slightly altered. In those experiments, MIU's accuracy was roughly equivalent to the figures reported here, although the improvement for all other query strategies (especially random selection) were lower.

## 5   Conclusion

We have presented *multiple-instance active learning*, a novel framework for reducing the labeling burden by obtaining labels at a coarse granularity, and then selectively labeling at finer levels. This approach is useful when bag labels are easily acquired, and instance labels can be obtained but are expensive. In the present work, we explored the case where an MI learner may query unlabeled instances from positively labeled bags in order reduce the inherent ambiguity of the MI representation, while keeping label costs low. We also described a simple method for learning from labels at both the bag-level and instance-level, and showed that querying instance-level labels through active learning is beneficial in content-based image retrieval and text categorization problems. In addition, we introduced two active query selection strategies motivated by this work, MI Uncertainty and Expected Gradient Length, and demonstrated that they are well-suited to MI active learning.

In future work, we plan to investigate the other MI active learning scenarios mentioned in Section 1. Of particular interest is the setting where, initially, some bags are labeled and others are not, and the learner is allowed to query on (i) unlabeled bags, (ii) unlabeled instances from positively labeled bags, or (iii) some combination thereof. We also plan to investigate other selection methods for different query formats, such as "label any or all positive instances in this bag," which may be more natural for some MI learning problems.

### Acknowledgments

This research was supported by NSF grant IIS-0093016 and NIH grants T15-LM07359 and R01-LM07050-05.

## Footnotes

[1]http://www.cs.wustl.edu/accio/

[2]http://people.csail.mit.edu/jrennie/20Newsgroups/

[3]http://pages.cs.wisc.edu/~bsettles/amil/

## References

[1] S. Andrews, I. Tsochantaridis, and T. Hofmann. Support vector machines for multiple-instance learning. In *Advances in Neural Information Processing Systems (NIPS)*, pages 561–568. MIT Press, 2003.

[2] D. Cohn, L. Atlas, and R. Ladner. Improving generalization with active learning. *Machine Learning*, 15(2):201–221, 1994.

[3] T. Dietterich, R. Lathrop, and T. Lozano-Perez. Solving the multiple-instance problem with axis-parallel rectangles. *Artificial Intelligence*, 89:31–71, 1997.

[4] J.T. Eppig, C.J. Bult, J.A. Kadin, J.E. Richardson, J.A. Blake, and the members of the Mouse Genome Database Group. The Mouse Genome Database (MGD): from genes to mice–a community resource for mouse biology. *Nucleic Acids Research*, 33:D471–D475, 2005. http://www.informatics.jax.org.

[5] D. Lewis and J. Catlett. Heterogeneous uncertainty sampling for supervised learning. In *Proceedings of the International Conference on Machine Learning (ICML)*, pages 148–156. Morgan Kaufmann, 1994.

[6] O. Maron and T. Lozano-Perez. A framework for multiple-instance learning. In *Advances in Neural Information Processing Systems (NIPS)*, pages 570–576. MIT Press, 1998.

[7] J. Nocedal and S.J. Wright. *Numerical Optimization*. Springer, 1999.

[8] R. Rahmani and S.A. Goldman. MISSL: Multiple-instance semi-supervised learning. In *Proceedings of the International Conference on Machine Learning (ICML)*, pages 705–712. ACM Press, 2006.

[9] S. Ray and M. Craven. Supervised versus multiple instance learning: An empirical comparison. In *Proceedings of the International Conference on Machine Learning (ICML)*, pages 697–704. ACM Press, 2005.

[10] Q. Tao, S.D. Scott, and N.V. Vinodchandran. SVM-based generalized multiple-instance learning via approximate box counting. In *Proceedings of the International Conference on Machine Learning (ICML)*, pages 779–806. Morgan Kaufmann, 2004.

[11] X. Zhu, J. Lafferty, and Z. Ghahramani. Combining active learning and semi-supervised learning using gaussian fields and harmonic functions. In *Proceedings of the ICML Workshop on the Continuum from Labeled to Unlabeled Data*, pages 58–65, 2003.
